# Binary Tuning is Optimal for Neural Rate Coding with High Temporal Resolution

**Matthias Bethge,* David Rotermund, and Klaus Pawelzik**
Institute of Theoretical Physics
University of Bremen
28334 Bremen
{mbethge,davrot,pawelzik}@physik.uni-bremen.de

## Abstract

Here we derive optimal gain functions for minimum mean square reconstruction from neural rate responses subjected to Poisson noise. The shape of these functions strongly depends on the length $T$ of the time window within which spikes are counted in order to estimate the underlying firing rate. A phase transition towards pure binary encoding occurs if the maximum mean spike count becomes smaller than approximately three provided the minimum firing rate is zero. For a particular function class, we were able to prove the existence of a second-order phase transition analytically. The critical decoding time window length obtained from the analytical derivation is in precise agreement with the numerical results. We conclude that under most circumstances relevant to information processing in the brain, rate coding can be better ascribed to a binary (low-entropy) code than to the other extreme of rich analog coding.

## 1 Optimal neuronal gain functions for short decoding time windows

The use of action potentials (spikes) as a means of communication is the striking feature of neurons in the central nervous system. Since the discovery by Adrian [1] that action potentials are generated by sensory neurons with a frequency that is substantially determined by the stimulus, the idea of rate coding has become a prevalent paradigm in neuroscience [2]. In particular, today the coding properties of many neurons from various areas in the cortex have been characterized by tuning curves, which describe the average firing rate response as a function of certain stimulus parameters. This way of description is closely related to the idea of analog coding, which constitutes the basis for many neural network models. Reliable inference from the observed number of spikes about the underlying firing rate of a neuronal response, however, requires a sufficiently long time interval, while integration times of neurons *in vivo* [3] as well as reaction times of humans or animals when performing classification tasks [4, 5] are known to be rather short. Therefore, it is important to understand, how neural rate coding is affected by a limited time window available for decoding.

While rate codes are usually characterized by tuning functions relating the intensity of the

neuronal response to a particular stimulus parameter, the question, how relevant the idea of analog coding actually is does not depend on the particular entity represented by a neuron. Instead it suffices to determine the shape of the gain function, which displays the mean firing rate as a function of the actual analog signal to be sent to subsequent neurons. Here we seek for optimal gain functions that minimize the minimum average squared reconstruction error for a uniform source signal transmitted through a Poisson channel as a function of the maximum mean number of spikes.

In formal terms, the issue is to optimally encode a real random variable $x$ in the number of pulses emitted by a neuron within a certain time window. Thereby, $x$ stands for the intended analog output of the neuron that shall be signaled to subsequent neurons. The latter, however, can only observe a number of spikes $k$ integrated within a time interval of length $T$. The statistical dependency between $x$ and $k$ is specified by the assumption of Poisson noise

$$p(k|\mu(x)) = \frac{(\mu(x))^k}{k!} \exp\{-\mu(x)\} ,\qquad (1)$$

and the choice of the gain function $f(x)$, which together with $T$ determines the mean spike count $\mu(x) = Tf(x)$. An important additional constraint is the limited output range of the neuronal firing rate, which can be included by the requirement of a bounded gain function ($f_{min} \leq f(x) \leq f_{max}$, $\forall\, x$). Since inhibition can reliably prevent a neuron from firing, we will here consider the case $f_{min} = 0$ only. Instead of specifying $f_{max}$, we impose a bound directly on the mean spike count (i.e. $\mu(x) \leq \bar{\mu}$), because $f_{max}$ constitutes a meaningful constraint only in conjunction with a fixed time window length $T$.

As objective function we consider the *minimum mean squared error* (MMSE) with respect to Lebesgue measure for $x \in [0,1]$,

$$\chi^2[\mu(x)] = E[x^2] - E[\hat{x}^2] = \frac{1}{3} - \sum_{k=0}^{\infty} \frac{\left(\int_0^1 x\, p(k|\mu(x))\, dx\right)^2}{\int_0^1 p(k|\mu(x))\, dx} ,\qquad (2)$$

where $\hat{x}(k) = E[x|k]$ denotes the *mean square estimator*, which is the conditional expectation (see e.g. [6]).

## 1.1 Tunings and errors

As derived in [7] on the basis of Fisher information the optimal gain function for a single neuron in the asymptotic limit $T \to \infty$ has a parabolic shape:

$$f^{asymp}(x) = f_{max}x^2 .\qquad (3)$$

For any finite $\bar{\mu}$, however, this gain function is not necessarily optimal, and in the limit $T \to 0$, it is straight forward to show that the optimal tuning curve is a step function

$$f^{step}(x|\vartheta) = f_{max}\, \Theta\, (x - \vartheta) ,\qquad (4)$$

where $\Theta(z)$ denotes the Heaviside function that equals one, if $z > 0$ and zero if $z < 0$. The optimal threshold $\vartheta(\bar{\mu})$ of the step tuning curve depends on $\bar{\mu}$ and can be determined analytically

$$\vartheta(\bar{\mu}) = 1 - \frac{3 - \sqrt{8e^{-\bar{\mu}} + 1}}{4(1 - e^{-\bar{\mu}})}\qquad (5)$$

as well as the corresponding MMSE [8]:

$$\chi^2[f^{step}] = \frac{1}{12}\left(1 - \frac{3\,\vartheta^2(\bar{\mu})}{[(1 - \vartheta(\bar{\mu}))(1 - e^{-\bar{\mu}})]^{-1} - 1}\right) .\qquad (6)$$

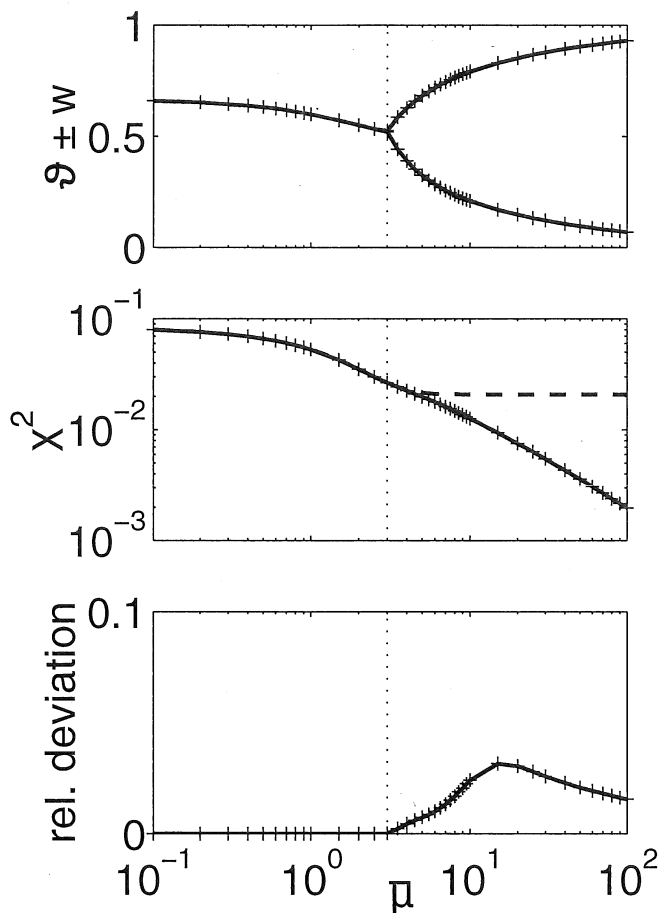

Figure 1: The upper panel shows a bifurcation plot for $\vartheta(\bar{\mu}) - w$ and $\vartheta(\bar{\mu}) + w$ of the optimal gain function in $\mathcal{S}_1$ as a function of $\bar{\mu}$ illustrating the phase transition from binary to continuous encoding. The dotted line separates the regions before and after the phase transition in all three panels. Left from this line (i.e. for $\bar{\mu} < \bar{\mu}^c$) the step function given by Eq. 4+5 is optimal. The middle panel shows the MMSE of this step function (dashed) and of the optimal gain function in $\mathcal{S}_2$ (solid), which becomes smaller than the first one after the phase transition. The relative deviation between the minimal errors of $\mathcal{S}_1$ and $\mathcal{S}_2$ (i.e. $(\chi_{\mathcal{S}_1}^2 - \chi_{\mathcal{S}_2}^2)/\chi_{\mathcal{S}_2}^2)$ is displayed in the lower panel and has a maximum below 0.035.

The binary shape for small $\bar{\mu}$ and the continuous parabolic shape for large $\bar{\mu}$ implies that there has to be a transition from discrete to analog encoding with increasing $\bar{\mu}$. Unfortunately it is not possible to determine the optimal gain function within the set of all bounded functions $\mathcal{B} := \{f | f : [0,1] \rightarrow [0, f_{max}]\}$ and hence, one has to choose a certain parameterized function space $\mathcal{S} \subset \mathcal{B}$ in advance that is feasible for the optimization. In [8], we investigated various such function spaces and for $\bar{\mu} < 2.9$, we did not find any gain function with an error smaller than the MMSE of the step function. Furthermore, we always observed a phase transition from binary to analog encoding at a critical $\bar{\mu}^c$ that depends only slightly on the function space. As one can see in Fig. 1 (upper) $\bar{\mu}^c$ is approximately three.

In this paper, we consider two function classes $\mathcal{S}_1, \mathcal{S}_2$, which both contain the binary gain function as well as the asymptotic optimal parabolic function as special cases. Furthermore $\mathcal{S}_1$ is a proper subset of $\mathcal{S}_2$. Our interest in $\mathcal{S}_1$ results from the fact that we can analyze the phase transition in this subset analytically, while $\mathcal{S}_2$ is the most general parameterization for which we have determined the optimal encoding numerically. The latter has six free parameters $a \leq b \leq c \in [0,1]$, $f_{mid} \in (0, f_{max})$, $\alpha, \beta \in [0, \infty)$ and the parameterization of the gain functions is given by

$$f^{\mathcal{S}_2}(x|a,b,c,f_{mid},\alpha,\beta) = \begin{cases} 0 & , \quad 0 < x < a \\ f_{mid}\left(\frac{x-a}{b-a}\right)^\alpha & , \quad a < x < b \\ f_{mid} + (f_{max} - f_{mid})\left(\frac{x-b}{c-b}\right)^\beta & , \quad b < x < c \\ f_{max} & , \quad c < x < 1 \end{cases} . \tag{7}$$

The integrals entering Eq. 2 for the MMSE in case of the gain function $f^{\mathcal{S}_2}$ then read

$$
\begin{aligned}
\int_0^1 x\, p(k|x)\, dx = \frac{1}{k!} \Bigg\{ &\frac{a^2}{2}\delta_{0,k} + \frac{(b-a)^2\, \Gamma_{0,f_{mid}}\left(k+\frac{2}{\alpha}\right)}{\alpha(\sqrt[\alpha]{f_{mid}})^2} \\
&+ \frac{a(b-a)\, \Gamma_{0,f_{mid}}\left(k+\frac{1}{\alpha}\right)}{\alpha \sqrt[\alpha]{f_{mid}}} \\
&+ \frac{(c-b)^2\, \Gamma_{f_{mid},f_{max}}\left(k+\frac{2}{\beta}\right)}{\beta(\sqrt[\beta]{f_{max}} - \sqrt[\beta]{f_{mid}})^2} \\
&+ \left(b - \frac{\sqrt[\beta]{f_{mid}}(c-b)}{(\sqrt[\beta]{f_{max}} - \sqrt[\beta]{f_{mid}})}\right) \frac{(c-b)\, \Gamma_{f_{mid},f_{max}}\left(k+\frac{1}{\beta}\right)}{\beta(\sqrt[\beta]{f_{max}} - \sqrt[\beta]{f_{mid}})} \\
&+ \frac{(1-c^2)}{2} f_{max}^k e^{-f_{max}} \Bigg\}
\end{aligned}
\tag{8}
$$

$$
\begin{aligned}
\int_0^1 p(k|x)\, dx = \frac{1}{k!} \Bigg\{ &a\delta_{0,k} + \frac{(b-a)\, \Gamma_{0,f_{mid}}\left(k+\frac{1}{\alpha}\right)}{\alpha \sqrt[\alpha]{f_{mid}}} \\
&+ \frac{(c-b)\, \Gamma_{f_{mid},f_{max}}\left(k+\frac{1}{\beta}\right)}{\beta(\sqrt[\beta]{f_{max}} - \sqrt[\beta]{f_{mid}})} + (1-c)f_{max}^k e^{-f_{max}} \Bigg\} ,
\end{aligned}
\tag{9}
$$

where $\Gamma_{u,v}(z) = \int_u^v s^{z-1} e^{-s}\, ds$ denotes the truncated Gamma function. Numerical optimization leads to the minimal MMSE as a function of $\bar{\mu}$ as displayed in Fig. 1 (middle). The parameterization of the gain functions in $\mathcal{S}_1$ is given by

$$f^{\mathcal{S}_1}(x|w,\gamma) = \begin{cases} 0 & , \quad 0 < x < \vartheta(\bar{\mu}) - w \\ f_{max}\left(\frac{x-\vartheta(\bar{\mu})+w}{2w}\right)^\gamma & , \quad \vartheta(\bar{\mu}) - w < x < \vartheta(\bar{\mu}) + w \\ f_{max} & , \quad \vartheta(\bar{\mu}) + w < x < 1 \end{cases} , \tag{10}$$

with $w \in [0,1]$ and $\gamma \in [0, \infty)$. The integrals entering Eq. 2 for the MMSE in case of the

gain function $f^{\mathcal{S}_1}$ read

$$
\begin{aligned}
\int_0^1 x\, p(k|x)\, dx &= \frac{1}{k!}\left\{ \frac{(\vartheta(\bar{\mu})-w)^2}{2}\delta_{0,k} + \frac{4w^2\Gamma_{0,f_{max}}\left(k+\frac{2}{\gamma}\right)}{\gamma(\sqrt[\gamma]{f_{max}})^2} \right. \\
&\quad + \frac{2w(\vartheta(\bar{\mu})-w)\Gamma_{0,f_{max}}\left(k+\frac{1}{\gamma}\right)}{\gamma\sqrt[\gamma]{f_{max}}} \\
&\quad \left. + \frac{1-(\vartheta(\bar{\mu})+w)^2}{2}f_{max}^k e^{-f_{max}} \right\}
\end{aligned}
\tag{11}
$$

$$
\begin{aligned}
\int_0^1 p(k|x)\, dx &= \frac{1}{k!}\left\{ (\vartheta(\bar{\mu})-w)\delta_{0,k} + \frac{2w\Gamma_{0,f_{max}}\left(k+\frac{1}{\gamma}\right)}{\gamma\sqrt[\gamma]{f_{max}}} \right. \\
&\quad \left. + (1-\vartheta(\bar{\mu})-w)f_{max}^k e^{-f_{max}} \right\}
\end{aligned}
\tag{12}
$$

The minimal MMSE for these gain functions is only slightly worse than that for $\mathcal{S}_2$. The relative difference between both is plotted in Fig. 1 (lower) showing a maximum deviation of 3.2%. In particular, the relative deviation is extremely small around the phase transition. This comparison suggests that a restriction to $\mathcal{S}_1$, which is a necessary simplification for the following analytical investigation, does not change the qualitative results.

## 2 A phase transition

The phase transition from binary to analog encoding corresponds to a structural change of the objective function $\chi^2(w,\gamma)$. In particular, the optimality of binary encoding for $\bar{\mu} < \bar{\mu}^c$ implies that $\chi^2(w,\gamma)$ has a minimum at $w = 0$. The existence of a phase transition implies that with increasing $\bar{\mu}$ this minimum changes into a local maximum at a certain critical point $\bar{\mu} = \bar{\mu}^c$. Therefore, the critical point can be determined by a local expansion of

$$
\chi^2(w,\gamma,\bar{\mu}) - \chi^2(0,\gamma,\bar{\mu}) = \sum_{k=1}^{\infty} g_k(\lambda,\bar{\mu})\frac{w^k}{k!}
\tag{13}
$$

around $w = 0$, because the sign of its leading coefficient $A_\gamma(\bar{\mu})$ (i.e. the coefficient $g_k$ with minimal $k$ that does not vanish identically) determines, whether $\chi^2(w,\gamma,\bar{\mu})$ has a local minimum or maximum at $w = 0$. Accordingly, the critical point is given as the solution of $A_\gamma(\bar{\mu}) = 0$.

With quite a bit of efforts one can prove that the first derivative of $\chi^2(w,\gamma,\bar{\mu})$ vanishes for all $\bar{\mu}$. The second derivative, however, is a decreasing function of $\bar{\mu}$ and hence constitutes the wanted leading coefficient

$$
\begin{aligned}
A_\gamma(\bar{\mu}) &= \frac{1}{4(e^{\bar{\mu}}-1)^2}\left\{ 8 - 7e^{\bar{\mu}} + 16e^{2\bar{\mu}} + e^{3\bar{\mu}} \right. \\
&\quad - \sqrt{1+8e^{-\bar{\mu}}}\left(2+e^{\bar{\mu}}\left(-3+e^{\bar{\mu}}(6+e^{\bar{\mu}})\right)\right) \\
&\quad + \left(16e^{\bar{\mu}} - 48e^{2\bar{\mu}} - 4e^{3\bar{\mu}} + \sqrt{1+8e^{-\bar{\mu}}}\left(4e^{\bar{\mu}} - 8\left(4+e^{\bar{\mu}}\right)\right)\right)\frac{\bar{\mu}^{-\frac{1}{\gamma}}}{\gamma}\Gamma_{0,\bar{\mu}}\left(\frac{1}{\gamma}\right) \\
&\quad + \left(8e^{2\bar{\mu}} + 2\left(5 - 3\sqrt{1+8e^{-\bar{\mu}}}\right)e^{3\bar{\mu}}\right)\frac{\bar{\mu}^{-\frac{2}{\gamma}}}{\gamma^2}\Gamma_{0,\bar{\mu}}^2\left(\frac{1}{\gamma}\right)
\end{aligned}
$$

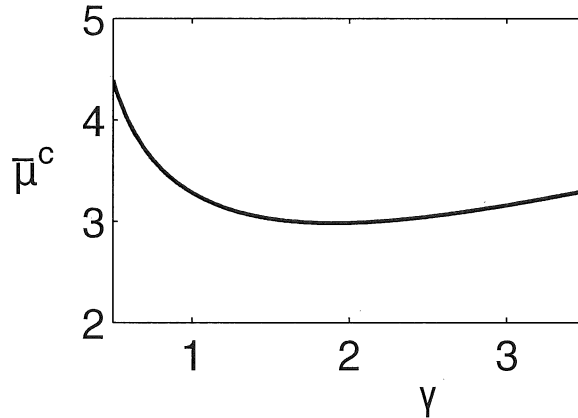

Figure 2: The critical maximum mean spike count $\mu^c$ is shown as a function of $\gamma$ (numerical evaluation at $\gamma \in \{0.5, 0.505, 0.51, \ldots, 3.5\}$). The minimum $\mu^c = 2.98291 \pm 10^{-7}$ at $\gamma = 1.9$ determines the phase transition in $\mathcal{S}_1$.

$$
\begin{aligned}
& - \quad 16e^{\bar{\mu}} \left( e^{\bar{\mu}} - 1 \right) \left( \sqrt{1 + 8e^{-\bar{\mu}}} - 3 \right) \frac{\bar{\mu}^{-\frac{2}{\gamma}}}{\gamma} \Gamma_{0,\bar{\mu}} \left( \frac{2}{\gamma} \right) \\
& + \quad 2e^{2\bar{\mu}} \left( e^{\bar{\mu}} - 1 \right) \left( \sqrt{1 + 8e^{-\bar{\mu}}} - 3 \right) \frac{\bar{\mu}^{-\frac{2}{\gamma}}}{\gamma^2} \\
& \cdot \quad \int_0^{\bar{\mu}} e^{-s} s^{\frac{1-\gamma}{\gamma}} \left( 1 - \frac{s}{\bar{\mu}} \right)^{-\frac{1}{\gamma}} \Gamma_{0,\bar{\mu}-s} \left( \frac{1}{\gamma} \right) ds \Bigg\} \quad .
\end{aligned}
\tag{14}
$$

Obviously, it is not possible to write the zeros of $A_\gamma(\bar{\mu})$ in a closed form. The numerical evaluation of the critical point $\bar{\mu}^c(\gamma)$ as a function of $\gamma$ is displayed in Fig. 2. Note, that we have treated $\gamma$ as a fixed parameter, which means that we determine the critical point of the phase transition in all subsets $\mathcal{S}_1(\gamma)$ of $\mathcal{S}_1$ that correspond to a fixed $\gamma$. It is straight forward to show that the critical point $\bar{\mu}^c$ with respect to the entire class $\mathcal{S}_1$ is given by the minimum of $\bar{\mu}^c(\gamma)$. We determined this value up to a precision of $\pm 0.0001$ to be $\bar{\mu}^c = 2.9857$.

## 3 Conclusion

Our study reveals that optimal encoding with respect to the minimum mean squared error is binary for maximum mean spike counts smaller than approximately three. Within the function class $\mathcal{S}_1$ we determined a second-order phase transition from binary to continuous encoding analytically. With respect to mutual information the advantage of binary encoding holds even up to a maximum mean spike count of about 3.5 (results not shown) and remains discrete also for larger $\bar{\mu}$. In a related work [9], Softky compared the information capacity of the Poisson channel with the information rate of a (noiseless) binary pulse code. The rate of the latter turned out to exceed the capacity of the former at a factor of at least 72 demonstrating a clear superiority of binary coding over analog rate coding. Our rate-distortion analysis of the Poisson channel differs from that comparison in a twofold way: First, we do not change the noise model and second, the MMSE is often more appropriate to account for the coding efficiency than the channel capacity [10]. In particular, the assumption of a real random variable to be encoded with minimal mean squared error loss appears to introduce a bias for analog coding rather than for binary coding. Nevertheless,

assuming a high temporal precision (i.e. small integration times $T$), our results hint into a similar direction, namely that binary coding seems to be a more reasonable choice even if one supposes that the only means of neuronal communication would be the transmission of Poisson distributed spike counts.

Methodologically, our analysis is similar to many theoretical studies of population coding if $f(x) = \mu(x)/T$ is not interpreted as the neuron's gain function, but as a tuning function with respect to a stimulus parameter $x$. Though conceptually different, some readers may therefore wish to know whether binary coding is still advantageous if many neurons, say $N$, together encode for a single analog value. While the approach chosen in this paper is not feasible in case of large $N$, a partial answer can be given: For the efficiency of population coding redundancy reduction is most important [7, 8, 11]. Smooth tuning curves, which have a dynamic range at about the same size as the signal range always lead to a large amount of redundancy so that the MMSE can not decrease faster than $N^{-1}$. In contrast the MMSE of binary tuning functions scales proportional to $N^{-2}$ or even faster. This holds also true for tuning functions, which are not perfectly binary, but have a dynamic range that is at least smaller than the signal range divided by $N$. Independent from $\bar{\mu}$ this implies that a small dynamic range is always advantageous in case of population coding.

In contrast, most experimental studies do not report on binary or steep tuning functions, but show smooth tuning curves only. However, the shape of a tuning function always depends on the stimulus set used. Only recently, experimental studies under natural stimulus conditions provided evidence for the idea that neuronal encoding is essentially binary [12]. Particularly striking is this observation for the H1 neuron of the fly [13], for which the functional role is probably better understood than for most other neurons that have been characterized by tuning functions.

While the noise level of the Poisson channel studied in this paper is rather large, the H1 neuron can respond very reliably under optimal stimulus conditions [13]. Another example of a low-noise binary code has been found in the auditory cortex [14]. If we drop the restriction to Poisson noise and impose a hard constraint on the maximum number of spikes instead, optimal encoding is always discrete with $\mu(x)$ taking integer values only [15]. This is easy to grasp, because any rational $\mu$ can not serve to increase the entropy of the available symbol set (i.e. the candidate spike counts), but only increases the noise entropy instead. In other words, it is the simple fact that spike counts are discrete by nature, which already severely limits the possibility of graded rate coding. Clearly, this is not so obvious in case of the Poisson channel, if there is no hard constraint imposed on the maximum spike count.

A remarkable aspect of the neuronal response of H1 shown in [13] is that it becomes the more binary the less noisy the stimulus conditions are (the noise level is determined by the different light conditions at midday, half an hour before, and half an hour after sunset). This suggests an interesting hypothesis why choosing a binary code with very high temporal precision might be advantageous even if the signal of interest by itself does not change at that time scale: the sensory input may sometimes be too noisy, so that repeated, independent samples from the signal of interest may sometimes lead to neuronal firing and sometimes not. In other words, a binary code at the short time scale is useful independent from the correlation time of the signal to be encoded, if uncertainties have to be taken into account, because any surplus available amount of temporal precision is maximally used for uncertainty representation in a self-adjusting manner. Furthermore, this Monte-Carlo type of uncertainty representation features several computational advantages [16]. Finally, it is a remarkable fact that this property is unique for a binary code, because the representation of uncertainty is necessary for many information processing tasks solved by the brain.

Additional support for the potential relevance of a binary neural code comes from intracellular recordings *in vivo* revealing that the subthreshold membrane potential of many cortical cells switches between up and down states [17] depending on the stimulus. Furthermore,

the dynamics of bursting cells plays an important role for neuronal signal transmission [18] and may also be seen as evidence for binary rate coding. In light of these experimental facts, we conclude from our results that the idea of binary tuning constitutes an important hypothesis for neural coding.

**Acknowledgments**

This work was supported by the Deutsche Forschungsgesellschaft SFB 517.

## Footnotes

*http://www.neuro.uni-bremen.de/~mbethge

# References

[1] E.D. Adrian. The impulses produced by sensory nerve endings: Part i. *J. Physiol. (London)*, 61:49–72, 1926.

[2] D.H. Perkel and T.H. Bullock. Neural coding: a report based on an nrp work session. *Neurosci. Research Prog. Bull.*, 6:220–349, 1968.

[3] W.R. Softky and C. Koch. The hihgly irregular firing of cortical cells is inconsistent with temporal integration of random epsps. *J. Neurosci.*, 13:334–350, 1993.

[4] C. Keysers, D. Xiao, P. Foldiak, and D. Perrett. The speed of sight. *J. Cog. Neurosci.*, 13:90–101, 2001.

[5] S. Thorpe, D. Fize, and Marlot. Speed of processing in the human visual system. *Nature*, 381:520–522, 1996.

[6] E.L. Lehmann and G. Casella. *Theory of point estimation*. Springer, New York, 1999.

[7] M. Bethge, D. Rotermund, and K. Pawelzik. Optimal short-term population coding: when fisher information fails. *Neural Comput.*, 14(10):2317–2351, 2002.

[8] M. Bethge, D. Rotermund, and K. Pawelzik. Optimal neural rate coding leads to bimodal firing rate distributions. *Network: Comput. Neural Syst.*, 2002. in press.

[9] W.R. Softky. Fine analog coding minimizes information transmission. *Neural Networks*, 9:15–24, 1996.

[10] D.H. Johnson. Point process models of single-neuron discharges. *J. Comput. Neurosci.*, 3:275–299, 1996.

[11] M. Bethge and K. Pawelzik. Population coding with unreliable spikes. *Neurocomputing*, 44-46:323–328, 2002.

[12] P. Reinagel. How do visual neurons respond in the real world. *Curr. Op. Neurobiol.*, 11:437–442, 2001.

[13] G.D. Lewen, W. Bialek, and R.R. de Ruyter van Steveninck. Neural coding of natural stimuli. *Network: Comput. Neural Syst.*, 12:317–329, 2001.

[14] M.R. DeWeese and A.M. Zador. Binary coding in auditory cortex. In S. Becker, S. Thrun, and K. Obermayer, editors, *Advances in Neural Information Processing Systems*, volume 15, 2002.

[15] A. Gersho and R.M. Grey. *Vector quantization and signal compression*. Kluwer, Boston, 1992.

[16] P.O. Hoyer and A. Hyvarinen. Interpreting neural response variability as monte carlo sampling of the posterior. In S. Becker, S. Thrun, and K. Obermayer, editors, *Advances in Neural Information Processing Systems*, volume 15, 2002.

[17] J. Anderson, I. Lampl, I. Reichova, M. Carandini, and D. Ferster. Stimulus dependence of two-state fluctuations of membrane potential in cat visual cortex. *Nature Neurosci.*, 3:617–621, 2000.

[18] J.E. Lisman. Bursts as a unit of neural information processing: making unreliable synapses reliable. *TINS*, 20:38–43, 1997.
